# Regularizing AdaBoost

**Gunnar Rätsch, Takashi Onoda,* Klaus R. Müller**
GMD FIRST, Rudower Chaussee 5, 12489 Berlin, Germany
{raetsch, onoda, klaus}@first.gmd.de

## Abstract

Boosting methods maximize a hard classification margin and are known as powerful techniques that do not exhibit overfitting for low noise cases. Also for noisy data boosting will try to enforce a hard margin and thereby give too much weight to outliers, which then leads to the dilemma of non-smooth fits and overfitting. Therefore we propose three algorithms to allow for soft margin classification by introducing regularization with slack variables into the boosting concept: (1) AdaBoost$_{reg}$ and regularized versions of (2) linear and (3) quadratic programming AdaBoost. Experiments show the usefulness of the proposed algorithms in comparison to another soft margin classifier: the support vector machine.

## 1   Introduction

Boosting and other ensemble methods have been used with success in several applications, e.g. OCR [13, 8]. For **low noise** cases several lines of explanation have been proposed as candidates for explaining the well functioning of boosting methods. (a) Breiman proposed that during boosting also a "bagging effect" takes place [3] which reduces the variance and effectively limits the capacity of the system and (b) Freund et al. [12] show that boosting classifies with large margins, since the error function of boosting can be written as a function of the margin and every boosting step tries to minimize this function by maximizing the margin [9, 11].

Recently, studies with **noisy** patterns have shown that boosting does indeed overfit on noisy data, this holds for boosted decision trees [10], RBF nets [11] and also other kinds of classifiers (e.g. [7]). So it is clearly a myth that boosting methods will not overfit. The fact that boosting is trying to maximize the margin, is exactly also the argument that can be used to understand why boosting must necessarily overfit for noisy patterns or overlapping distributions and we give asymptotic arguments for this statement in section 3. Because the hard margin (smallest margin in the trainings set) plays a central role in causing overfitting, we propose to relax the hard margin classification and allow for misclassifications by using the soft margin classifier concept that has been applied to support vector machines successfully [5].

Our view is that the margin concept is central for the understanding of both support vector machines and boosting methods. So far it is not clear what the optimal margin distribution should be that a learner has to achieve for optimal classification in the noisy case. For data without noise a hard margin might be the best choice. However, for noisy data there is always the trade-off in believing in the data or mistrusting it, as the very data point could be an outlier. In general (e.g. neural network) learning strategies this leads to the introduction of regularization which reflects the prior that we have about a problem. We will also introduce a regularization strategy (analogous to weight decay) into boosting. This strategy uses slack variables to achieve a soft margin (section 4). Numerical experiments show the validity of our regularization approach in section 5 and finally a brief conclusion is given.

## 2   AdaBoost Algorithm

Let $\{h_t(\mathbf{x}) : t = 1, \ldots, T\}$ be an ensemble of $T$ hypotheses defined on input vector $\mathbf{x}$ and $\mathbf{c} = [c_1 \ldots c_T]$ their weights satisfying $c_t > 0$ and $|\mathbf{c}| = \sum_t c_t = 1$. In the binary classification case, the output is one of two class labels, i.e. $h_t(\mathbf{x}) = \pm 1$.
The ensemble generates the label which is the weighted majority of the votes: $\text{sgn}\left(\sum_t c_t h_t(\mathbf{x})\right)$. In order to train this ensemble of $T$ hypotheses $\{h_t(\mathbf{x})\}$ and $\mathbf{c}$, several algorithms have been proposed: bagging, where the weighting is simply $c_t = 1/T$ [2] and AdaBoost/Arcing, where the weighting scheme is more complicated [12]. In the following we give a brief description of AdaBoost/Arcing. We use a special form of Arcing, which is equivalent to AdaBoost [4]. In the binary classification case we define the margin for an input-output pair $\mathbf{z}_i = (\mathbf{x}_i, y_i), i = 1, \ldots, l$ by

$$mg(\mathbf{z}_i, \mathbf{c}) = y_i \sum_{t=1}^{T} c_t h_t(\mathbf{x}_i), \tag{1}$$

which is between $-1$ and $1$, if $|\mathbf{c}| = 1$. The correct class is predicted, if the margin at $\mathbf{z}$ is positive. When the positivity of the margin value increases, the decision correctness becomes larger. AdaBoost maximizes the margin by (asymptotically) minimizing a function of the margin $mg(\mathbf{z}_i, \mathbf{c})$ [9, 11]

$$g(\mathbf{b}) = \sum_{i=1}^{l} \exp\left\{-\frac{|\mathbf{b}|}{2} mg(\mathbf{z}_i, \mathbf{c})\right\}, \tag{2}$$

where $\mathbf{b} = [b_1 \ldots b_T]$ and $|\mathbf{b}| = \sum_t b_t$ (starting from $\mathbf{b} = 0$). Note that $b_t$ is the unnormalized weighting of the hypothesis $h_t$, whereas $\mathbf{c}$ is simply a normalized version of $\mathbf{b}$, i.e. $\mathbf{c} = \mathbf{b}/|\mathbf{b}|$. In order to find the hypothesis $h_t$ the learning examples $\mathbf{z}_i$ are weighted in each iteration $t$ with $w_t(\mathbf{z}_i)$. Using a bootstrap on this weighted sample we train $h_t$; alternatively a weighted error function can be used (e.g. weighted MSE). The weights $w_t(\mathbf{z}_i)$ are computed according to[1]

$$w_t(\mathbf{z}_i) = \frac{\exp\left\{-|\mathbf{b}_{t-1}| mg(\mathbf{z}_i, \mathbf{c}_{t-1})/2\right\}}{\sum_{j=1}^{l} \exp\left\{-|\mathbf{b}_{t-1}| mg(\mathbf{z}_j, \mathbf{c}_{t-1})/2\right\}} \tag{3}$$

and the training error $\epsilon_t$ of $h_t$ is computed as $\epsilon_t = \sum_{i=1}^{l} w_t(\mathbf{z}_i) I(y_i \neq h_t(\mathbf{x}_i))$, where $I(true) = 1$ and $I(false) = 0$. For each given hypothesis $h_t$ we have to find a weight $b_t$, such that $g(\mathbf{b})$ is minimized. One can optimize this parameter by a line search

or directly by analytic minimization [4], which gives $b_t = \log(1 - \epsilon_t) - \log \epsilon_t$. Interestingly, we can write

$$w_t(\mathbf{z}_i) = \frac{\partial g(\mathbf{b}_{t-1})/\partial mg(\mathbf{z}_i, \mathbf{b}_{t-1})}{\sum_{j=1}^{l} \partial g(\mathbf{b}_{t-1})/\partial mg(\mathbf{z}_j, \mathbf{b}_{t-1})}, \tag{4}$$

as a gradient of $g(\mathbf{b}_{t-1})$ with respect to the margins. The weighted minimization with $w_t(\mathbf{z}_i)$ will give a hypothesis $h_t$ which is an approximation to the best possible hypothesis $h_t^*$ that would be obtained by minimizing $g$ directly. Note that, the weighted minimization (bootstrap, weighted LS) will not necessarily give $h_t^*$, even if $\epsilon_t$ is minimized [11]. AdaBoost is therefore an *approximate* gradient descent method which minimizes $g$ asymptotically.

## 3 Hard margins

A decrease of $g(\mathbf{c}, |\mathbf{b}|) := g(\mathbf{b})$ is predominantly achieved by improvements of the margin $mg(\mathbf{z}_i, \mathbf{c})$. If the margin $mg(\mathbf{z}_i, \mathbf{c})$ is negative, then the error $g(\mathbf{c}, |\mathbf{b}|)$ takes clearly a big value, which is additionally amplified by $|\mathbf{b}|$. So, AdaBoost tries to decrease the negative margin efficiently to improve the error $g(\mathbf{c}, |\mathbf{b}|)$.

Now, let us consider the asymptotic case, where the number of iterations and therefore also $|\mathbf{b}|$ take large values [9]. In this case, when the values of all $mg(\mathbf{z}_i, \mathbf{c}), i = 1, \cdots, l$, are almost the same but have small differences, these differences are amplified strongly in $g(\mathbf{c}, |\mathbf{b}|)$. Obviously the function $g(\mathbf{c}, |\mathbf{b}|)$ is asymptotically very sensitive to small differences between margins. Therefore, the margins $mg(\mathbf{z}_i, \mathbf{c})$ of the training patterns from the margin area (boundary area between classes) should asymptotically converge to the same value. From Eq. (3), when $|\mathbf{b}|$ takes a very big value, AdaBoost learning becomes a "hard competition" case: only the pattern with smallest margin will get high weights, the other patterns are effectively neglected in the learning process. In order to confirm that the above reasoning is correct, Fig. 1 shows margin distributions after $10^4$ AdaBoost iterations for a toy example [9] at different noise levels generated by uniform distribution $U(0.0, \sigma^2)$ (left). From this figure, it becomes apparent that the margin distribution asymptotically makes a step at a fixed size of the margin for training patterns which are in the margin area. In previous studies [9, 11] we observed that those patterns exhibit a large overlap to support vectors in support vector machines. The numerical results support our theoretical asymptotic analysis. The property of AdaBoost to produce a big margin area (no pattern in the area, i.e. a hard margin), will not always lead to the best generalization ability (cf. [5, 11]). This is especially true,

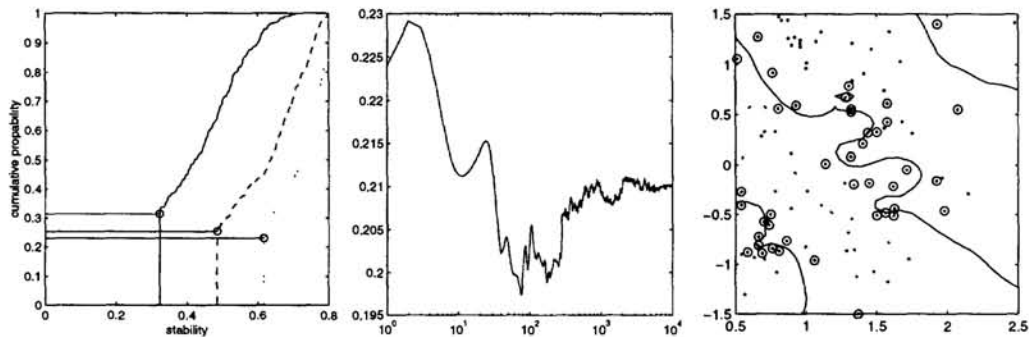

Figure 1: Margin distributions for AdaBoost (left) for different noise levels ($\sigma^2 = 0\%$(dotted), 9%(dashed), 16%(solid)) with fixed number of RBF-centers for the base hypothesis and typical overfitting behaviour in the generalization error as a function of the number of iterations (middle) and a typical decision line (right) generated by AdaBoost using RBF networks in the case with noise (here: 30 centers and $\sigma^2 = 16\%$; smoothed)

if the training patterns have classification or input noise. In our experiments with noisy data, we often observed that AdaBoost made overfitting (for a high number of boosting iterations). Fig. 1 (middle) shows a typical overfitting behaviour in the generalization error for AdaBoost: after only 80 boosting iterations the best generalization performance is already achieved. Quinlan [10] and Grove et al. [7] also observed overfitting and that the generalization performance of AdaBoost is often worse than that of the single classifier, if the data has classification noise.

The first reason for overfitting is the increasing value of $|\mathbf{b}|$: noisy patterns (e.g. bad labelled) can asymptotically have an "unlimited" influence to the decision line leading to overfitting (cf. Eq. (3)). Another reason is the classification with a hard margin, which also means that all training patterns will asymptotically be correctly classified (without any capacity limitation!). In the presence of noise this will certainly be not the right concept, because the best decision line (e.g. Bayes) usually will not give a training error of zero. So, the achievement of large hard margins for noisy data will produce hypotheses which are too complex for the problem.

## 4    How to get Soft Margins

**Changing AdaBoost's error function**    In order to avoid overfitting, we introduce slack variables, which are similar to those of the support vector algorithm [5, 14], into AdaBoost.

We know that all training patterns will get non-negative stabilities after many iterations(see Fig. 1(left)), i.e. $mg(\mathbf{z}_i, \mathbf{c}) \geq \rho$ for all $i = 1, \ldots, l$, where $\rho$ is the minimum margin of the patterns. Due to this fact, AdaBoost often produces high weights for the difficult training patterns by enforcing a non-negative margin $\rho \geq 0$ (for every pattern including outliers) and this property will eventually lead to overfitting, as observed in Fig. 1. Therefore, we introduce some variables $\xi_i$ - the slack variables - and get

$$mg(\mathbf{z}_i, \mathbf{c}) \geq \rho - C\xi_i^t, \qquad \xi_i^t > 0. \tag{5}$$

In these inequalities, $\xi_i^t$ are positive and if a training pattern has high weights in the previous iterations, the $\xi_i^t$ should be increasing. In this way, for example, we do not force outliers to be classified according to their possibly wrong labels, but we allow for some errors. In this sense we get a trade-off between the margin and the importance of a pattern in the training process (depending on the constant $C \geq 0$). If we choose $C = 0$ in Eq. (5), the original AdaBoost algorithm is retrieved. If $C$ is chosen too high, the data is not taken seriously. We adopt a prior on the weights $w_r(\mathbf{z}_i)$ that punishes large weights in analogy to weight decay and choose

$$\xi_i^t = \left( \sum_{r=1}^{t} c_r w_r(\mathbf{z}_i) \right)^2, \tag{6}$$

where the inner sum is the cumulative weight of the pattern in the previous iterations (we call it *influence* of a pattern – similar to Lagrange multipliers in SVMs). By this $\xi_i^t$, AdaBoost is not changed for easy classifiable patterns, but is changed for difficult patterns. From Eq. (5), we can derive a new error function:

$$\tilde{g}_{reg}(\mathbf{c}_t, |\mathbf{b}_t|) = \sum_{i=1}^{l} \exp \left\{ -\frac{|\mathbf{b}_t|}{2} mg(\mathbf{z}_i, \mathbf{c_t}) - C\xi_i^t \right\} \tag{7}$$

By this error function, we can control the trade-off between the weights, which the pattern had in the last iterations, and the achieved margin. The weight $w_t(\mathbf{z}_i)$ of a pattern is computed as the derivative of Eq. (7) subject to $mg(\mathbf{z}_i, \mathbf{b}^{t-1})$ (cf. Eq. (4)) and is given by

$$w_t(\mathbf{z}_i) = \frac{\exp\left\{ |\mathbf{b}_{t-1}|(mg(\mathbf{z}_i, c_{t-1}) - \xi_i^{t-1})/2 \right\}}{\sum_{j=1}^{l} \exp\left\{ |\mathbf{b}_{t-1}|(mg(\mathbf{z}_j, c_{t-1}) - \xi_j^{t-1})/2 \right\}}. \tag{8}$$

Table 1: Pseudocode description of the algorithms

**LP-AdaBoost(Z, T)**  |  **LP$_{reg}$-AdaBoost(Z, T, C)**  |  **QP$_{reg}$-AdaBoost(Z, T, C)**

Run AdaBoost on dataset **Z** to get $T$ hypotheses **h** and their weights **c**

$$\text{Construct loss matrix } L_{i,t} = \begin{cases} -1 & \text{if } h_t(\mathbf{x}_i) \neq y_i \\ 1 & \text{otherwise} \end{cases}$$

| minimize $-\rho$ | minimize $-\rho + C\sum_i \xi_i$ | minimize $\|\mathbf{b}\|^2 + C\sum_i \xi_i$ |
|---|---|---|
| s.t. $\sum_{t=1}^T c_t L_{i,t} \geq \rho$ | s.t. $\sum_{t=1}^T c_t L_{i,t} \geq \rho + \xi_i$ | s.t. $\sum_{t=1}^T b_t L_{i,t} \geq 1 - \xi_i$ |
| $c_t \geq 0, \sum c_t = 1$ | $c_t \geq 0, \sum c_t = 1$ | $b_t \geq 0$ |
| | $\xi_i \geq 0$ | $\xi_i \geq 0$ |

Thus we can get an update rule for the weight of a training pattern [11]

$$w_t(\mathbf{z}_i) = w_{t-1}(\mathbf{z}_i)\exp\{b_{t-1}I(y_i \neq h_{t-1}(\mathbf{x}_i)) + C\xi_i^{t-2}|\mathbf{b}_{t-2}| - C\xi_i^{t-1}|\mathbf{b}_{t-1}|\}. \quad (9)$$

It is more difficult to compute the weight $b_t$ of the $t$-th hypothesis analytically. However, we can get $b_t$ by a line search procedure over Eq. (7), which has an unique solution because $\frac{\partial}{\partial b_t}g_{reg} > 0$ is satisfied. This line search can be implemented very efficiently. With this line-search, we can now also use real-valued outputs of the base hypotheses, while the original AdaBoost algorithm could not (cf. also [6]).

**Optimizing a given ensemble**     In Grove et al. [7], it was shown how to use linear programming to maximize the minimum margin for a given ensemble and LP-AdaBoost was proposed (table 1 left). This algorithm maximizes the minimum margin on the training patterns. It achieves a hard margin (as AdaBoost asymptotically does) for small number of iterations. For the reasoning for a hard margin (section 3) this can not generalize well. If we introduce slack variables to LP-AdaBoost, one gets the algorithm LP$_{reg}$-AdaBoost (table 1 middle) [11]. This modification allows that some patterns have lower margins than $\rho$ (especially lower than 0). There is a trade-off: (a) make all margins bigger than $\rho$ and (b) maximize $\rho$. This trade-off is controlled by the constant $C$.

Another formulation of a optimization problem can be derived from the support vector algorithm. The optimization objective of a SVM is to find a function $h^{\mathbf{w}}$ which minimizes a functional of the form $E = \|\mathbf{w}\|^2 + C\sum_i \xi_i$, where $y_i h(\mathbf{x}_i) \geq 1 - \xi_i$ and the norm of the parameter vector $\mathbf{w}$ is the measure for the complexity of the hypothesis $h^{\mathbf{w}}$ [14]. For ensemble learning we do not have such a measure of complexity and so we use the norm of the hypotheses weight vector **b**. For $|\mathbf{b}| = 1$ this is a small value, if the elements are approximately equal (analogy to bagging) and has high values, when there are some strongly emphasized hypotheses (far away from bagging). Experimentally, we found that $\|\mathbf{b}\|^2$ is often larger for more complex hypothesis. Thus, we can apply the optimization principles of SVMs to AdaBoost and get the algorithm QP$_{reg}$-AdaBoost (table 1 right). We effectively use a linear SVM on top of the results of the base hypotheses.

## 5   Experiments

In order to evaluate the performance of our new algorithms, we make a comparison among the single RBF classifier, the original AdaBoost algorithm, AdaBoost$_{reg}$ (with RBF nets), L/QP$_{reg}$-AdaBoost and a Support Vector Machine (with RBF kernel). We use ten artificial and real world datasets from the UCI and DELVE benchmark repositories: banana (toy dataset as in [9, 11]), breast cancer, image segment, ringnorm, flare sonar, splice, new-thyroid, titanic, twonorm, waveform. Some of the problems are originally not binary classification problems, hence a (random) partition into two classes was used. At first we generate 20 partitions into training and test set (mostly $\approx 60\% : 40\%$). On each partition we train the classifier and get its test set error. The performance is averaged and we get table 2.

Table 2: Comparison among the six methods: Single RBF classifier, AdaBoost(AB), AdaBoost$_{reg}$(AB$_{reg}$), L/QP$_{reg}$-AdaBoost (L/QPR) and a Support Vector Machine(SVM): Estimation of generalization error in % on 10 datasets (best method in bold face). Clearly, AdaBoost$_{reg}$ gives the best overall performance. For further explanation see text.

|          | RBF      | AB       | AB$_{reg}$   | LPR      | QPR      | SVM      |
|----------|----------|----------|----------|----------|----------|----------|
| Banana   | 10.9±0.5 | 12.3±0.7 | **10.7±0.5** | 10.8±0.4 | 10.9±0.5 | 11.5±4.7 |
| Cancer   | 28.7±5.3 | 30.5±4.5 | 26.3±4.3 | 31.0±4.2 | 26.2±4.7 | **26.1±4.8** |
| Image    | 2.8±0.7  | 2.5±0.7  | 2.5±0.7  | 2.6±0.6  | **2.4±0.5** | 2.9±0.7  |
| Ringnorm | **1.7±0.3** | 2.0±0.2  | **1.7±0.2** | 2.2±0.4  | 1.9±0.2  | **1.7±0.1** |
| FSonar   | 34.6±2.1 | 35.6±1.9 | 33.6±1.7 | 35.7±4.5 | 36.2±1.7 | **32.5±1.7** |
| Splice   | 10.0±0.3 | 10.1±0.3 | **9.5±0.2** | 10.2±1.6 | 10.1±0.5 | 10.9±0.7 |
| Thyroid  | 4.8±2.4  | **4.4±1.9** | **4.4±2.1** | **4.4±2.0** | **4.4±2.2** | 4.8±2.2  |
| Titanic  | 23.4±1.7 | 22.7±1.2 | 22.5±1.0 | 22.9±1.9 | 22.7±1.0 | **22.4±1.0** |
| Twonorm  | 2.8±0.2  | 3.1±0.3  | **2.7±2.1** | 3.4±0.6  | 3.0±0.3  | 3.0±0.2  |
| Waveform | 10.7±1.0 | 10.8±0.4 | 9.9±0.9  | 10.6±1.0 | 10.1±0.5 | **9.8±0.3** |
| Mean %   | 6.7      | 9.6      | 1.0      | 11.1     | 4.7      | 6.3      |
| Winner % | 16.4     | 8.2      | 28.5     | 15.0     | 15.3     | 16.6     |

We used RBF nets with adaptive centers (some conjugate gradient iterations to optimize positions and widths of the centers) as base hypotheses as described in [1, 11]. In all experiments, we combined 200 hypotheses. Clearly, this number of hypotheses may be not optimal, however Adaboost with optimal early stopping is not better than AdaBoost$_{reg}$. The parameter $C$ of the regularized versions of AdaBoost and the parameters $(C, \sigma)$ of the SVM are optimized by the first five training datasets. On each training set 5-fold-cross validation is used to find the best model for this dataset[2]. Finally, the model parameters are computed as the median of the five estimations. This way of estimating the parameters is surely not possible in practice, but will make this comparison more robust and the results more reliable. The last but one line in Tab. 2 shows the line 'Mean %', which is computed as follows: For each dataset the average error rate of all classifier types are divided by the minimum error rate and 1 is subtracted. These resulting numbers are averaged over the 10 datasets. The last line shows the probabilities that a method wins, i.e. gives the smallest generalization error, on the basis of our experiments (averaged over all ten datasets). Our experiments on noisy data show that (a) the results of AdaBoost are in almost all cases worse than the single classifier (clear overfitting effect) and (b) the results of AdaBoost$_{reg}$ are in all cases (much) better than those of AdaBoost and better than that of the single classifier. Furthermore, we see clearly, that (c) the single classifier wins as often as the SVM, (d) L/QP$_{reg}$-AdaBoost improves the results of AdaBoost, (e) AdaBoost$_{reg}$ wins most often. L/QP$_{reg}$-AdaBoost improves the results of AdaBoost in almost cases due to established the soft margin. But the results are not as good as the results of AdaBoost$_{reg}$ and the SVM, because the hypotheses generated by AdaBoost (aimed to construct a hard margin) may be not the appropriate ones generate a good soft margin. We also observe that quadratic programming gives slightly better results than linear programming. This may be due to the fact that the hypotheses coefficients generated by LP$_{reg}$-AdaBoost are more sparse (smaller ensemble). Bigger ensembles may have a better generalization ability (due to the reduction of variance [3]). The worse performance of SVM compared to AdaBoost$_{reg}$ and the unexpected tie between SVM and RBF net may be explained with (a) the fixed $\sigma$ of the RBF-kernel (loosing multi-scale information), (b) coarse model selection, (c) worse error function of the SV algorithm (noise model). Sumarizing, AdaBoost is useful for low noise cases, where the classes are separable (as shown for OCR[13, 8]). AdaBoost$_{reg}$ extends the applicability of boosting to "difficult separable" cases and should be applied, if the data is noisy.

## 6    Conclusion

We introduced three algorithms to alleviate the overfitting problems of boosting algorithms for high noise data: (1) direct incorporation of the regularization term into the error function (Eq.(7)), use of (2) linear and (3) quadratic programming with constraints given by the slack variables. The essence of our proposal is to introduce slack variables for regularization in order to allow for soft margin classification in contrast to the hard margin classification used before. The slack variables basically allow to control how much we trust the data, so we are permitted to ignore outliers which would otherwise have spoiled our classification. This generalization is very much in the spirit of support vector machines that also trade-off the maximization of the margin and the minimization of the classification errors in the slack variables. In our experiments, $AdaBoost_{reg}$ showed a better overall generalization performance than all other algorithms including the Support Vector Machines. We conjecture that this unexpected result is mostly due to the fact that SVM can only use one $\sigma$ and therefore loose scaling information. AdaBoost does not have this limitation.

So far we balance our trust in the data and the margin maximization by cross validation. Better would be, if we knew the "optimal" margin distribution that we could achieve for classifying noisy patterns, then we could of course balance the errors and the margin sizes optimally.

In future works, we plan to establish more connections between AdaBoost and SVM.

**Acknowledgements:** We thank for valuable discussions with A. Smola, B. Schölkopf, T. Frieß and D. Schuurmans. Partial funding from EC STORM project grant number 25387 is greatfully acknowledged. The breast cancer domain was obtained from the University Medical Centre, Inst. of Oncology, Ljubljana, Yugoslavia. Thanks go to M. Zwitter and M. Soklic for providing the data.

## Footnotes

*permanent address: Communication & Information Research Lab. CRIEPI, 2-11-1 Iwado kita, Komae-shi, Tokyo 201-8511, Japan.

[1]This direct way for computing the weights is equivalent to the update rule of AdaBoost.

[2]The parameters are only near-optimal. Only 10 values for each parameter are tested.

## References

[1]  C. M. Bishop. *Neural Networks for Pattern Recognition.* Clarendon, 1995.

[2]  L. Breiman. Bagging predictors. *Machine Learning*, 26(2):123–140, 1996.

[3]  L. Breiman. Arcing classifiers. Tech.Rep. 460, Berkeley Stat.Dept., 1997.

[4]  L. Breiman. Prediction games and arcing algorithms. Tech.Rep. 504, Berkeley Stat.Dept., 1997.

[5]  C. Cortes, V. Vapnik. Support vector network. *Mach.Learn.*, 20:273–297, 1995.

[6]  R. Schapire, Y. Singer. Improved Boosting Algorithms Using Confidence-rated Predictions. In *Proc. of COLT'98*.

[7]  A.J. Grove, D. Schuurmans. Boosting in the limit: Maximizing the margin of learned ensembles. In *Proc. 15th Nat. Conf. on AI*, 1998. To appear.

[8]  Y. LeCun et al. Learning algorithms for classification: A comparism on handwritten digit recognistion. *Neural Networks*, pages 261–276, 1995.

[9]  T. Onoda, G. Rätsch, and K.-R. Müller. An asymptotic analysis of adaboost in the binary classification case. In *Proc. of ICANN'98*, April 1998.

[10]  J. Quinlan. Boosting first-order learning. In *Proc. of the 7th Internat. Workshop on Algorithmic Learning Theory, LNAI*, 1160, 143–155. Springer.

[11]  G. Rätsch. Soft Margins for AdaBoost. August 1998. Royal Holloway College, Technical Report NC-TR-1998-021. Submitted to Machine Learning.

[12]  R. Schapire, Y. Freund, P. Bartlett, W. Lee. Boosting the margin: A new explanation for the effectiveness of voting methods. *Mach.Learn.*, 148–156, 1998.

[13]  H. Schwenk and Y. Bengio. ˙Adaboosting neural networks: Application to online character recognition. In *ICANN'97*, LNCS, 1327, 967–972, 1997. Springer.

[14]  V. Vapnik. *The Nature of Statistical Learning Theory.* Springer, 1995.